# Evaluation of Rarity of Fingerprints in Forensics

**Chang Su and Sargur Srihari**
Department of Computer Science and Engineering
University at Buffalo
Amherst, NY 14260
{changsu,srihari}@buffalo.edu

## Abstract

A method for computing the rarity of latent fingerprints represented by minutiae is given. It allows determining the probability of finding a match for an evidence print in a database of $n$ known prints. The probability of random correspondence between evidence and database is determined in three procedural steps. In the *registration step* the latent print is aligned by finding its core point; which is done using a procedure based on a machine learning approach based on Gaussian processes. In the *evidence probability evaluation step* a generative model based on Bayesian networks is used to determine the probability of the evidence; it takes into account both the dependency of each minutia on nearby minutiae and the confidence of their presence in the evidence. In the *specific probability of random correspondence step* the evidence probability is used to determine the probability of match among $n$ for a given tolerance; the last evaluation is similar to the birthday correspondence probability for a specific birthday. The generative model is validated using a goodness-of-fit test evaluated with a standard database of fingerprints. The probability of random correspondence for several latent fingerprints are evaluated for varying numbers of minutiae.

## 1 Introduction

In many forensic domains it is necessary to characterize the degree to which a given piece of evidence is unique. For instance in the case of DNA a probability statement is made after a match has been confirmed between the evidence and the known, that the chance that a randomly selected person would have the same DNA pattern is 1 in 24,000,000 which is a description of rarity of the evidence/known [1]. In the case of fingerprint evidence there is uncertainty at two levels: the similarity between the evidence and the known and the rarity of the known. This paper explores the evaluation of the rarity of a fingerprint as characterized by a given set of features. Recent court challenges have highlighted the need for statistical research on this problem especially if it is stated that a high degree of similarity is present between the evidence and the known [2].

A statistical measure of the weight of evidence in forensics is a likelihood ratio (LR) defined as follows [3]. It is the ratio between the joint probability that the evidence and known come from the same source, and the joint probability that the two come from two different sources. If the underlying distributions are Gaussian the LR can be simplified as the product of two exponential factors: the first is a significance test of the null hypothesis of identity, and the second measures rarity. Since evaluation of the joint probability is difficult for fingerprints, which are characterized by variable sets of minutia points with each point itself expressed as a 3-tuple of spatial co-ordinates and an angle, the LR computation is usually replaced by one wherein a similarity (or kernel) function is introduced between the evidence and the known and the likelihood ratio is computed for the similarity [4, 5]. While such efforts concern the significance of the null hypothesis of identity, fingerprint rarity continues to be a difficult problem and has never been solved. This paper describes a systematic approach for the computation of the rarity of fingerprints in a robust and reliable manner.

The process involves several individual steps. Due to varying quality of fingerprints collected from the crime scene, called latent prints, a registration process is needed to determine which area of finger skin the print comes from; Section 2 describes the use of Gaussian processes to predict core points by which prints can be aligned. In Section 3 a generative model based on Bayesian networks is proposed to model the distribution of minutiae as well as the dependencies between them. To measure rarity, a metric for assessing the probability of random correspondence of a specific print against $n$ samples is defined in Section 4. The model is validated using a goodness-of-fit test in Section 5. Some examples of evaluation of rarity are given in Section 6.

## 2  Fingerprint Registration

The fingerprint collected from the crime scene is usually only a small portion of the complete fingerprint. So the feature set extracted from the print only contains relative spatial relationship. It's obvious that feature sets with same relative spatial relationship can lead to different rarity if they come from the different areas of the fingertip. To solve this problem, we first predict the core points and then align the fingerprints by overlapping their core points. In biometrics and fingerprint analysis, core point refers to the center area of a fingerprint. In practice, the core point corresponds to the center of the north most loop type singularity. For fingerprints that do not contain loop or whorl singularities, the core is usually associated with the point of maxima ridge line curvature[6]. The most popular approaches proposed for core point detection is the Poincare Index (PI) which is developed by [7, 8, 9]. Another commonly used method [10] is a sine map based method that is realized by multi-resolution analysis. The methods based on Fourier expansion[11], fingerprint structures [12] and multi-scale analysis [13] are also proposed. All of these methods require that the fingerprints are complete and the core points can be seen in the prints. But this is not the case for all the fingerprints. Latent prints are usually small partial prints and do not contain core points. So there's no way to detect them by above computational vision based approaches.

We proposes a core point prediction approach that turns this problem into a regression problem. Since the ridge flow directions reveal the intrinsic features of ridge topologies, and thus have critical impact on core point prediction. The orientation maps are used to predict the core points. A fingerprint field orientation map is defined as a collection of two-dimensional direction fields. It represents the directions of ridge flows in regular spaced grids. The gradients of gray intensity of enhanced fingerprints are estimated to obtain reliable ridge orientation [9]. Given an orientation map of a fingerprint, the core point is predicted using Gaussian processes. Gaussian processes dispense with the parametric model and instead define a probability distribution over functions directly. It provides more flexibility and better prediction. The advantage of Gaussian process model also comes from the probabilistic formulation[14]. Instead of representing the core point as a single value, the predication of the core point from Gaussian process model takes the form of a full predictive distribution.

Suppose we have a training set $\mathcal{D}$ of $N$ fingerprints, $\mathcal{D} = \{(\mathbf{g}_i, y_i) | i = 1, \ldots, N\}$, where $\mathbf{g}$ denotes the orientation map of a fingerprint print and $y$ denotes the output which is the core point. In order to predict the core points, Gaussian process model with squared exponential covariance function is applied. The regression model with Gaussian noise is given by

$$y = f(\mathbf{g}) + \epsilon \tag{1}$$

where $f(\mathbf{g})$ is the value of the process or function $f(x)$ at $\mathbf{g}$ and $\epsilon$ is a random noise variable whose value is chosen independent for each observation. We consider the noise processes that have a Gaussian distribution, so that the Gaussian likelihood for core point is given by

$$p(\mathbf{y}|f(\mathbf{g})) = \mathcal{N}(\mathbf{f}, \sigma^2 I) \tag{2}$$

where $\sigma^2$ is the variance of the noise. From the definition of a Gaussian process, the Gaussian process prior is given by a Gaussian whose mean is zero and whose covariance is defined by a covariance function $k(\mathbf{g}, \mathbf{g}')$ so that

$$f(\mathbf{g}) \sim \mathcal{GP}(0, k(\mathbf{g}, \mathbf{g}')) \tag{3}$$

The squared exponential covariance function is used here to specify the covariance between pairs of variables, parameterized by $\theta_1$ and $\theta_2$.

$$k(\mathbf{g}, \mathbf{g}') = \theta_1 \exp(-\frac{\theta_2}{2}|\mathbf{g} - \mathbf{g}'|^2) \tag{4}$$

where the hyperparameters $\theta_1$ and $\theta_2$ are optimized by maximizing of the log likelihood $p(\mathbf{y}|\theta_1, \theta_2)$

Suppose the orientation map of a input fingerprint is given by $\mathbf{g}^*$. The Gaussian predictive distribution of core point $y^*$ can be evaluated by conditioning the joint Gaussian prior distribution on the observation $(G, \mathbf{y})$, where $G = (\mathbf{g}_1, \ldots, \mathbf{g}_N)^\top$ and $\mathbf{y} = (y_1, \ldots, y_N)^\top$. The predictive distribution is given by

$$p(y^*|\mathbf{g}^*, G, \mathbf{y}) = \mathcal{N}(m(y^*), cov(y^*)) \tag{5}$$

where

$$m(y^*) = \mathbf{k}(\mathbf{g}^*, G)[K + \sigma^2 I]^{-1}\mathbf{y} \tag{6}$$

$$cov(y^*) = k(\mathbf{g}^*, \mathbf{g}^*) + \sigma^2 - \mathbf{k}(\mathbf{g}^*, G)^\top [K + \sigma^2 I]^{-1}\mathbf{k}(G, \mathbf{g}^*) \tag{7}$$

where $K$ is the Gram matrix whose elements are given by $k(\mathbf{g}_i, \mathbf{g}_j)$.

Note that for some fingerprints such as latent fingerprints collected from crime scene, their locations in the complete print are unknown. So any $\mathbf{g}^*$ only represents the orientation map of the print in one possible location. In order to predict the core point in the correct location, we list all the possible print locations corresponding to the different translations and rotations. The orientation maps of them are defined as $\mathcal{G} = \{\mathbf{g}_i^*|i = 1, \ldots, m\}$. Using (5), we obtain the predictive distributions $p(y^*|\mathbf{g}_i^*, G, \mathbf{y})$ for all the $\mathbf{g}_i^*$. The core point $\hat{y}^*$ should maximize $p(y^*|\mathbf{g}_i^*, G, \mathbf{y})$ with respect to $\mathbf{g}_i^*$. Thus the core point of the fingerprint is given by

$$\hat{y}^* = \mathbf{k}(\mathbf{g}_{MAX}^*, G)[K + \sigma^2 I]^{-1}\mathbf{y} \tag{8}$$

where $\mathbf{g}_{MAX}^*$ is the orientation map where the maximum predictive probability of core point can be obtained, given by

$$\mathbf{g}_{MAX}^* = \underset{\mathbf{g}^*}{\operatorname{argmax}}\, p(m(y^*)|\mathbf{g}^*, G, \mathbf{y}) \tag{9}$$

After the core points are determined, the fingerprints can be aligned by overlapping their core points. This is done by presenting the features in the Cartesian coordinates where the origin is the core point. Note that the minutia features mentioned in following sections have been aligned first.

## 3 A Generative Model for Fingerprints

In order to estimate rarity, statistical models need to be developed to represent the distribution of fingerprint features. Previous generative models for fingerprints involve different assumptions: uniform distribution of minutia locations and directions [15] and minutiae are independent of each other [16, 17]. However, minutiae that are spatially close tend to have similar directions with each other [18]. Moreover, fingerprint ridges flow smoothly with very slow orientation change. The variance of the minutia directions in different regions of the fingerprint are dependent on both their locations and location variance [19, 20]. These observations on the dependency between minutiae need to be accounted for in eliciting reliable statistical models. The proposed model incorporates the distribution of minutiae and the dependency relationship between them.

Minutiae are the most commonly used features for representing fingerprints. They correspond to ridge endings and ridge bifurcations. Each minutia is represented by its location and direction. The direction is determined by the ridge at the location. Automatic fingerprint matching algorithms use minutiae as the salient features [21], since they are stable and are reliably extracted. Each minutia is represented as $\mathbf{x} = (s, \theta)$ where $s = (x_1, x_2)$ is its location and $\theta$ its direction.

In order to capture the distribution of minutiae as well as the dependencies between them, we first propose a method to define a unique sequence for a given set of minutiae. Suppose that a fingerprint contains $N$ minutiae. The sequence starts with the minutia $\mathbf{x}_1$ whose location is closest to the core point. Each remaining minutia $\mathbf{x}_n$ is the spatially closest to the centroid defined by the arithmetic mean of the location coordinates of all the previous minutiae $\mathbf{x}_1, \ldots \mathbf{x}_{n-1}$. Given this sequence, the fingerprint can be represented by a minutia sequence $\mathbf{X} = (\mathbf{x}_1, \ldots, \mathbf{x}_N)$. The sequence is robust to the variance of the minutiae because the next minutia is decided by the all the previous minutiae. Given the observation that spatially closer minutiae are more strongly related, we only model the dependence between $\mathbf{x}_n$ and its nearest minutia among $\{\mathbf{x}_1, \ldots, \mathbf{x}_{n-1}\}$. Although not all the dependence is taken into account, this is a good trade-off between model accuracy and computational complexity. Figure 1(a) presents an example where $\mathbf{x}_5$ is determined because its distance to the centroid of $\{\mathbf{x}_1, \ldots, \mathbf{x}_4\}$ is minimal. Figure 1(b) shows the minutia sequence and the minutia

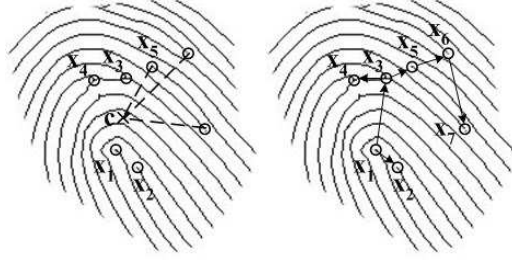

(a) Minutiae sequencing.  (b) Minutiae dependency.

Figure 1: Minutia dependency modeling: (a) given minutiae $\{\mathbf{x}_1, \ldots, \mathbf{x}_4\}$ with centroid $c$, the next minutia $\mathbf{x}_5$ is the one closest to $c$, and (b) following this procedure dependency between seven minutiae are represented by arrows.

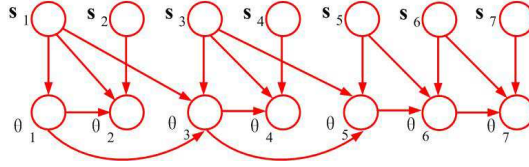

Figure 2: Bayesian network representing conditional dependencies shown in Figure 1, where $\mathbf{x}_i = (\mathbf{s}_i, \theta_I)$. Note that there is a link between $\mathbf{x}_1$ and $\mathbf{x}_2$ while there is none between $\mathbf{x}_2$ and $\mathbf{x}_3$.

dependencies (arrows) for the same configuration of minutiae. Based on the characteristic of fingerprint minutiae studied in [18, 19, 20], we know that the minutia direction is related to its location and the neighboring minutiae. The minutia location is conditional independent of the location of the neighboring minutiae given their directions. To address the probabilistic relationships of the minutiae, Bayesian networks are used to represent the distributions of the minutia features in fingerprints. Figure 2 shows the Bayesian network for the distribution of the minutia set given in Figure 1. The nodes $\mathbf{s}_n$ and $\theta_n$ represent the location and direction of minutia $\mathbf{x}_n$. For each conditional distribution, a directed link is added to the graph from the nodes corresponding to the variables on which the distribution is conditioned. In general, for a given fingerprint, the joint distribution over its minutia set $\mathbf{X}$ is given by

$$p(\mathbf{X}) = p(\mathbf{s}_1)p(\theta_1|\mathbf{s}_1) \prod_{n=2}^{N} p(\mathbf{s}_n)p(\theta_n|\mathbf{s}_n, \mathbf{s}_{\psi(n)}, \theta_{\psi(n)}) \tag{10}$$

where $\mathbf{s}_{\psi(n)}$ and $\theta_{\psi(n)}$ are the location and direction of the minutia $\mathbf{x}_i$ which has the minimal spatial distance to the minutia $\mathbf{x}_n$. So $\psi(n)$ is given by

$$\psi(n) = \underset{i \in [1, n-1]}{\operatorname{argmin}} \|\mathbf{x}_n - \mathbf{x}_i\| \tag{11}$$

To compute above joint probability, there are three probability density functions need to be estimated: distribution of the location of minutiae $f(\mathbf{s})$, joint distribution of the location and direction of minutiae $f(\mathbf{s}, \theta)$, and conditional distribution of minutia direction given its location, and the location and direction of the nearest minutia $f(\theta_n|\mathbf{s}_n, \mathbf{s}_{\psi(n)}, \theta_{\psi(n)})$.

It is known that minutiae tend to form clusters [18] and minutiae in different regions of the fingerprint are observed to be associated with different region-specific minutia directions. A mixture of Gaussian is a natural approach to model the minutia location given by (12). Since minutia orientation is a periodic variable, it is modeled by the von Mises distribution which itself is derived from the Gaussian. The minutia represented by its location and direction is modeled by the mixture of joint Gaussian and von-Mises distribution [22] give by (13). Given its location and the nearest minutia, the minutia direction has the mixture of von-Mises density given by (14).

$$f(\mathbf{s}) = \sum_{k_1=1}^{K_1} \pi_{k_1} \mathcal{N}(\mathbf{s}|\mu_{k_1}, \Sigma_{k_1}) \tag{12}$$

$$f(\mathbf{s}, \theta) = \sum_{k_2=1}^{K_2} \pi_{k_2} \mathcal{N}(\mathbf{s}|\mu_{k_2}, \Sigma_{k_2}) \mathcal{V}(\theta|\nu_{k_2}, \kappa_{k_2}) \tag{13}$$

$$f(\theta_n|\mathbf{s}_n, \mathbf{s}_{\psi(n)}, \theta_{\psi(n)}) = \sum_{k_3=1}^{K_3} \pi_{k_3} \mathcal{V}(\theta_n|\nu_{k_3}, \kappa_{k_3}) \tag{14}$$

where $K_i$ is the number of mixture components, $\pi_{k_i}$ are non-negative component weights that sum to one, $\mathcal{N}(s|\mu_k, \Sigma_k)$ is the bivariate Gaussian probability density function of minutiae with mean $\mu_k$ and covariance matrix $\Sigma_k$, and $\mathcal{V}(\theta|\nu_k, \kappa_k)$ is the von-Mises probability density function of minutia orientation with mean angle $\nu_k$ and precision (inverse variance) $\kappa_{k_3}$. Bayesian information criterion is used to estimate $K_i$ and other parameters are learned by EM algorithm.

## 4 Evaluation of Rarity of a Fingerprint

The general probability of random correspondence (PRC) can be modified to give the probability of matching the specific evidence within a database of $n$ items, where the match is within some tolerance in feature space [23]. The metric of rarity is specific nPRC, the probability that data with value $\mathbf{x}$ coincides with an element in a set of $n$ samples, within specified tolerance. Since we are trying to match a specific value $\mathbf{x}$, this probability depends on the probability of $\mathbf{x}$. Let $\mathbf{Y} = [\mathbf{y}_1, ..., \mathbf{y}_n]$ represent a set of $n$ random variables. A binary-valued random variable $z$ indicates that if one sample $\mathbf{y}_i$ exists in a set of $n$ random samples so that the value of $\mathbf{y}_i$ is the same as $\mathbf{x}$ within a tolerance $\epsilon$. By noting the independence of $\mathbf{x}$ and $\mathbf{y}_i$, the specific nPRC is then given by the marginal probability

$$p(z=1|\mathbf{x}) = \sum_{\mathbf{Y}} p(z=1|\mathbf{x}, \mathbf{Y}) p(\mathbf{Y}) \tag{15}$$

where $p(\mathbf{Y})$ is the joint probability of the n individuals.

To compute specific nPRC, we first define correspondence or match, between two minutiae as follows. Let $\mathbf{x}_a = (\mathbf{s}_a, \theta_a)$ and $\mathbf{x}_b = (\mathbf{s}_b, \theta_b)$ be a pair of minutiae. The minutiae are said to correspond if for tolerance $\epsilon = [\epsilon_s, \epsilon_\theta]$,

$$\| \mathbf{s}_a - \mathbf{s}_b \| \leq \epsilon_s \wedge |\theta_a - \theta_b| \leq \epsilon_\theta \tag{16}$$

where $\|\mathbf{s}_a - \mathbf{s}_b\|$ is the Euclidean distance between the minutia locations. Then, the match between two fingerprints is defined as existing at least $\hat{m}$ pairs of matched minutiae between two fingerprints. The tolerances $\epsilon$ and $\hat{m}$ depend on practical applications.

To deal with the largely varying quality in latent fingerprints, it is also important to consider the minutia confidence in specific nPRC measurement. The confidence of the minutia $\mathbf{x}_n$ is defined as $(d_{s_n}, d_{\theta_n})$, where $d_{s_n}$ is the confidence of location and $d_{\theta_n}$ is the confidence of direction. Given the minutia $\mathbf{x}_n = (s_n, \theta_n)$ and its confidences, the probability density functions of location $s'$ and direction $\theta'$ can be modeled using Gaussian and von-Mises distribution given by

$$c(s'|s_n, d_{s_n}) = \mathcal{N}(s'|s_n, d_{s_n}^{-1}) \tag{17}$$

$$c(\theta'|\theta_n, d_{\theta_n}) = \mathcal{V}(\theta'|\theta_n, d_{\theta_n}) \tag{18}$$

where the variance of the location distribution (Gaussian) is the inverse of the location confidence and the concentration parameter of the direction distribution (von-Mises) is the direction confidence.

Let $f$ be a randomly sampled fingerprint which has minutia set $\mathbf{X}' = \{\mathbf{x}'_1, ..., \mathbf{x}'_M\}$. Let $\widetilde{\mathbf{X}}$ and $\widetilde{\mathbf{X}'}$ be the sets of $\hat{m}$ minutiae randomly picked from $\mathbf{X}$ and $\mathbf{X}'$, where $\hat{m} \leq N$ and $\hat{m} \leq M$. Using (10), the probability that there is a one-to-one correspondence between $\widetilde{\mathbf{X}}$ and $\widetilde{\mathbf{X}'}$ is given by

$$p_\epsilon(\widetilde{\mathbf{X}}) = p_\epsilon(\mathbf{s}_1, \theta_1) \prod_{n=2}^{\hat{m}} p_\epsilon(\mathbf{s}_n) p_\epsilon(\theta_n|\mathbf{s}_n, \mathbf{s}_{\psi(n)}, \theta_{\psi(n)}) \tag{19}$$

where

$$p_\epsilon(\mathbf{s}_n, \theta_n) = \int_{s'} \int_{\theta'} \iint_{|\mathbf{x}-\mathbf{x}'|\leq\epsilon} c(\mathbf{s}'|\mathbf{s}_n, d_{s_n}) c(\theta'|\theta_n, d_{\theta_n}) f(\mathbf{s}, \theta) ds' d\theta' d\mathbf{s} d\theta \tag{20}$$

$$p_\epsilon(\mathbf{s}_n) = \int\limits_{s'} \int\limits_{|\mathbf{s}-\mathbf{s}'|\leq\epsilon_\mathbf{s}} c(\mathbf{s}'|\mathbf{s}_n, d_{s_n})f(\mathbf{s})ds'd\mathbf{s} \tag{21}$$

$$p_\epsilon(\theta_n|\mathbf{s}_n, \mathbf{s}_{\psi(n)}, \theta_{\psi(n)}) = \int\limits_{\theta'} \int\limits_{|\theta-\theta'|\leq\epsilon_\theta} c(\theta'|\theta_n, d_{\theta_n})f(\theta|\mathbf{s}_n, \mathbf{s}_{\psi(n)}, \theta_{\psi(n)})d\theta'd\theta \tag{22}$$

Finally, the specific $n$PRCs can be computed by

$$p_\epsilon(\mathbf{X}, \hat{m}, n) = 1 - (1 - p_\epsilon(\mathbf{X}, \hat{m}))^{n-1} \tag{23}$$

where $\mathbf{X}$ represents the minutia set of given fingerprint, and $p_\epsilon(\mathbf{X}, \hat{m})$ is the probability that $\hat{m}$ pairs of minutiae are matched between the given fingerprint and a randomly chosen fingerprint from $n$ fingerprints.

$$p_\epsilon(\mathbf{X}, \hat{m}) = \sum_{m'\in M} p(m')\binom{m'}{\hat{m}} \cdot \sum_{i=1}^{\binom{N}{\hat{m}}} p_\epsilon(\widetilde{\mathbf{X}}_i) \tag{24}$$

where $M$ contains all possible numbers of minutiae in one fingerprint among $n$ fingerprints, $p(m')$ is the probability of a random fingerprint having $m'$ minutiae, minutia set $\widetilde{\mathbf{X}}_i = (\mathbf{x}_{i1}, \mathbf{x}_{i2}, ..., \mathbf{x}_{i\hat{m}})$ is the subset of $\mathbf{X}$ and $p_\epsilon(\widetilde{\mathbf{X}}_i)$ is the joint probability of minutia set $\widetilde{\mathbf{X}}_i$ given by (19). Gibbs sampling is used to approximate the integral involved in the probability calculation.

## 5   Model Validation

In order to validate the proposed methods, core point prediction was first tested. Goodness-of-fit tests were performed on the proposed generative models. Two databases were used, one is NIST4, and the other is NIST27. NIST4 contains 8-bit gray scale images of randomly selected fingerprints. Each print has $512 \times 512$ pixels. The entire database contains fingerprints taken from 2000 different fingers with 2 impression of the same finger. NIST27 contains latent fingerprints from crime scenes and their matching rolled fingerprint mates. There are 258 latent cases separated into three quality categories of good, bad, and ugly.

### 5.1   Core Point Prediction

The Gaussian process models for core point prediction are trained on NIST4 and tested on NIST27. The orientation maps are extracted by conventional gradient-based approach. The fingerprint images are first divided into equal-sized blocks of $N \times N$ pixels, where $N$ is the average width of a pair of ridge and valley. The value of $N$ is 8 in NIST4 and varies in NIST27. The gradient vectors are calculated by taking the partial derivatives of image intensity at each pixel in Cartesian coordinates. The ridge orientation is perpendicular to the dominant gradient angle in the local block. The training set consists of the orientation maps of the fingerprints and the corresponding core points which are marked manually. The core point prediction is applied on three groups of latent prints in different quality. Figure 3 shows the results of core point prediction and subsequent latent print localization given two latent fingerprints from NIST27. Table 1 shows the comparison of prediction precisions of Gaussian Processes (GP) based approach and the widely used Poincare Index (PI) [8]. The test latent prints are extracted and enhanced manually. The true core points of the latent prints are picked from the matching 10-prints. Correct prediction is determined by comparing the location and direction distances between predicted and true core points with the threshold parameters set at $T_s = 16$ pixels, and $T_\theta = \pi/6$. Good quality set contains 88 images that mostly contain the core points. Both bad and ugly quality sets contain 85 images that have small size and usually do not include core points. Among the precisions of good quality latent prints, two approaches are close. Precisions of bad and ugly quality show distinct difference between two methods and indicate that GP based method provides core point prediction even though the core points can not be seen in the latent prints. The GP based method also results in higher overall prediction precisions.

### 5.2   Goodness-of-fit

The validation of the proposed generative model is by means of a goodness-of-fit test which determines as to how well a sample of data agrees with the proposed model distribution. The chi-square

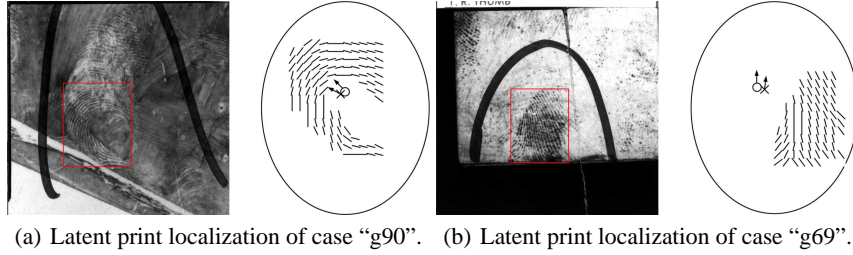

(a) Latent print localization of case "g90".    (b) Latent print localization of case "g69".

Figure 3: Latent print localization: Left side images are the latent fingerprints (rectangles) collected from crime scenes. Right side images contain the predicted core points (crosses) and true core points (rounds) with the orientation maps of the latent prints.

Table 1: Comparison of prediction precisions of PI and GP based approaches.

|          | Poincare Index | Gaussian Processes |
|----------|----------------|--------------------|
| Good     | 90.6%          | 93.1%              |
| Bad      | 68.2%          | 87.1%              |
| Ugly     | 46.6%          | 72.7%              |
| Overall  | 68.6%          | 84.5%              |

statistical hypothesis test was applied [24]. Three different tests were conducted for : (i) distribution of minutia location (12), (ii) joint distribution of minutia location and orientation (13), and (iii) distributions of minutia dependency (14). For minutia location, we partitioned the minutia location space into 16 non-overlapping blocks. For minutia location and orientation, we partitioned the feature space into $16 \times 4$ non-overlapping blocks. For minutia dependency, the orientation space is divided into 9 non-overlapping blocks. The blocks are combined with adjacent blocks until both observed and expected numbers of minutiae in the block are greater than or equal to 5. The test statistic used here is a chi-square random variable $\chi^2$ defined by the following equation.

$$\chi^2 = \sum_i \frac{(O_i - E_i)^2}{E_i} \tag{25}$$

where $O_i$ is the observed minutia count for the $i$th block, and $E_i$ is the expected minutia count for the $i$th block. The $p$-value, the probability of observing a sample statistic as extreme as the test statistic, associated with each test statistic $\chi^2$ is then calculated based on the chi-square distribution and compared to the significance level. For the NIST 4 dataset, we chose significance level equal to 0.01. 4000 fingerprints are used to train the generative models proposed in Sections 3.

To test the models for minutia location, and minutia location and orientation, the numbers of fingerprints with $p$-values above (corresponding to accept the model) and below (corresponding to reject the model) the significance level are computed. Of the 4000 fingerprints, 3387 are accepted and 613 are rejected for minutia location model, and 3216 are accepted and 784 are rejected for minutia location and orientation model. To test the model for minutia dependency, we first collect all the linked minutia pairs in the minutia sequences produced from 4000 fingerprints. Then these minutia pairs are separated by the binned locations of both minutiae ($32 \times 32$) and orientation of the leading minutia (4). Finally, the minutia dependency models can be tested on the corresponding minutia pair sets. Of the 4096 data sets, 3558 are accepted and 538 are rejected. The results imply that the proposed generative models offer reasonable and accurate fit to fingerprints.

Table 2: Results from the Chi-square tests for testing the goodness of fit of three generative models.

| Generative models | Dataset sizes | Model accepted | Model rejected |
|-------------------|---------------|----------------|----------------|
| $f(\mathbf{s})$ | 4000 | 3387 | 613 |
| $f(\mathbf{s}, \theta)$ | 4000 | 3216 | 784 |
| $f(\theta_n \mid \mathbf{s}_n, \mathbf{s}_{\psi(n)}, \theta_{\psi(n)})$ | 4096 | 3558 | 538 |

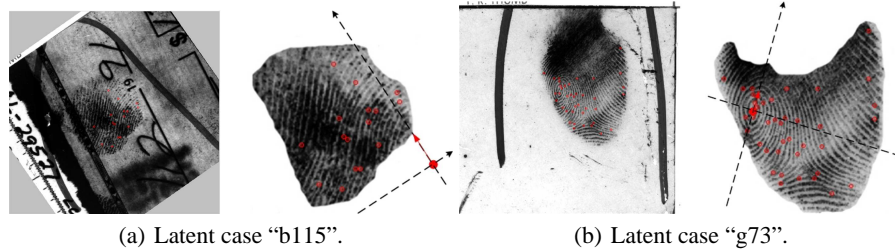

<div align="center">(a) Latent case "b115".       (b) Latent case "g73".</div>

Figure 4: Two latent cases: The left images are the crime scene photographs containing the latent fingerprints and minutiae. The right images are the preprocessed latent prints with aligned minutiae with predicted core points.

Table 3: Specific $n$PRCs for the latent fingerprints "b115" and "g73", where $n = 100,000$.

| Latent Print "b115" | | | Latent Print "g73" | | |
|---|---|---|---|---|---|
| $N$ | $\hat{m}$ | $p_\epsilon(\hat{m}, \mathbf{X})$ | $N$ | $\hat{m}$ | $p_\epsilon(\hat{m}, \mathbf{X})$ |
| | 2 | 0.73 | | 4 | 1 |
| | 4 | $9.04 \times 10^{-6}$ | | 8 | $3.11 \times 10^{-14}$ |
| 16 | 8 | $2.46 \times 10^{-19}$ | 39 | 12 | $2.56 \times 10^{-25}$ |
| | 12 | $6.13 \times 10^{-31}$ | | 24 | $3.10 \times 10^{-52}$ |
| | 16 | $1.82 \times 10^{-46}$ | | 39 | $7.51 \times 10^{-79}$ |

# 6  Fingerprint Rarity measurement on Latent Prints

The method for assessing fingerprint rarity using the validated model is demonstrated here. Figure 4 shows two latent fingerprints randomly picked from NIST27. The first latent print "b115" contains 16 minutiae and the second "g73" contains 39 minutiae. The confidences of minutiae are manually assigned by visual inspection. The specific $n$PRC of the two latent prints are given by Table 3. The specific $n$PRCs are calculated through varying numbers of matching minutia pairs ($\hat{m}$), assuming that the number of fingerprints ($n$) is $100,000$. The tolerance is set at $\epsilon_s = 10$ pixels and $\epsilon_\theta = \pi/8$.

The experiment shows that the values of specific $n$PRC are largely dependent on the given latent fingerprint. For the latent print that contains more minutiae or whose minutiae are more common in minutia population, the probability that the latent print shares $\hat{m}$ minutiae with a random fingerprint is more. It is obvious to note that, when $\hat{m}$ decreases, the probability of random correspondence increases. Moreover, the values of specific $n$PRC provide a strong argument for the values of latent fingerprint evidences.

# 7  Summary

This work is the first attempt of offering a systematic method to measure the rarity of fingerprints. In order to align the prints, a Gaussian processes based approach is proposed to predict the core points. It is proven that this approach can predict core points whether the prints contain the core points or not. Furthermore, a generative model is proposed to model the distribution of minutiae as well as the dependency between them. Bayesian networks are used to perform inference and learning by visualizing the structures of the generative models. Finally, the rarity of a fingerprint is able to calculated. To further improve the accuracy, minutia confidences are taken into account for specific $n$PRC calculation. Goodness of fit tests shows that the proposed generative offers an accurate fingerprint representation. We perform the specific $n$PRC computation on NIST27 dataset. It is shown that the proposed method is capable of estimating the rarity of real-life latent fingerprints.

**Acknowledgments**

This work was supported by the United States Department of Justice award NIJ: 2009-DN-BX-K208. The opinions expressed are those of the authors and not of the DOJ.

# References

[1] R. Chakraborty. Statistical interpretation of DNA typing data. *American Journal of Human Genetics*, 49(4):895–897, 1991.

[2] United States Court of Appeals for the Third Circuit: USA v. Byron Mitchell, 2003. No. 02-2859.

[3] D.V. Lindley. A problem in forensic science. *Biometrika*, 64(2):207–213, 1977.

[4] C. Neumann, C. Champod, R. Puch-Solis, N. Egli, A. Anthonioz, and A. Bromage-Griffiths. Computation of likelihood ratios in fingerprint identification for configurations of any number of minutiae. *Journal of Forensic Sciences*, 51:1255–1266, 2007.

[5] S.N. Srihari and H. Srinivasan. Comparison of ROC and Likelihood Decision Methods in Automatic Fingerprint Verification. *International J. Pattern Recognition and Artificial Intelligence*, 22(1):535–553, 2008.

[6] A.K. Jain and D. Maltoni. *Handbook of Fingerprint Recognition*. Springer-Verlag New York, Inc., Secaucus, NJ, USA, 2003.

[7] M. Kawagoe and A. Tojo. Fingerprint pattern classification. *Pattern Recogn.*, 17(3):295–303, 1984.

[8] A.M. Bazen and S.H. Gerez. Systematic methods for the computation of the directional fields and singular points of fingerprints. *IEEE Trans. Pattern Anal. Mach. Intell.*, 24(7):905–919, 2002.

[9] A.K. Jain, S. Prabhakar, and L. Hong. A multichannel approach to fingerprint classification. *IEEE Trans. Pattern Anal. Mach. Intell.*, 21(4):348–359, 1999.

[10] A.K. Jain, S. Prabhakar, L. Hong, and S. Pankanti. Filterbank-based fingerprint matching. *IEEE Transactions on Image Processing*, 9:846–859, 2000.

[11] D. Phillips. A fingerprint orientation model based on 2d fourier expansion (fomfe) and its application to singular-point detection and fingerprint indexing. *IEEE Trans. Pattern Anal. Mach. Intell.*, 29(4):573–585, 2007.

[12] X. Wang, J. Li, and Y. Niu. Definition and extraction of stable points from fingerprint images. *Pattern Recogn.*, 40(6):1804–1815, 2007.

[13] M. Liu, X. Jiang, and A.C. Kot. Fingerprint reference-point detection. *EURASIP J. Appl. Signal Process.*, 2005:498–509, 2005.

[14] C.E. Rasmussen and C.K.I. Williams. *Gaussian Processes for Machine Learning*. the MIT Press, 2006.

[15] S. Pankanti, S. Prabhakar, and A.K. Jain. On the individuality of fingerprints. *IEEE Trans. Pattern Anal. Mach. Intell.*, 24(8):1010–1025, 2002.

[16] Y. Zhu, S.C. Dass, and A.K. Jain. Statistical models for assessing the individuality of fingerprints. *IEEE Transactions on Information Forensics and Security*, 2(3-1):391–401, 2007.

[17] Y. Chen and A.K. Jain. Beyond minutiae: A fingerprint individuality model with pattern, ridge and pore features. In *ICB '09 Proceedings*, pages 523–533, Berlin, Heidelberg, 2009. Springer-Verlag.

[18] S.C. Scolve. The occurence of fingerprint characteristics as a two dimensional process. *Journal of the American Statistical Association*, 367(74):588–595, 1979.

[19] D.A. Stoney. Distribution of epidermal ridge minutiae. *American Journal of Physical Anthropology*, 77:367–376, 1988.

[20] J. Chen and Y. Moon. A statistical study on the fingerprint minutiae distribution. In *ICASSP 2006 Proceedings.*, volume 2, pages II–II, 2006.

[21] C. Watson, M. Garris, E. Tabassi, C. Wilson, R. McCabe, and S. Janet. *User's Guide to NIST Fingerprint Image Software 2 (NFIS2)*. NIST, 2004.

[22] C. Bishop. *Pattern Recognition and Machine Learning*. Springer, New York, 2006.

[23] C. Su and S.N. Srihari. Probability of random correspondence for fingerprints. In *IWCF '09 Proceedings*, pages 55–66, Berlin, Heidelberg, 2009. Springer-Verlag.

[24] R.B. D'Agostino and M.A. Stephens. *Goodness-of-fit Techniques*. CRC Press, 1986.

